# HETEROGENEOUS NEURAL NETWORKS FOR ADAPTIVE BEHAVIOR IN DYNAMIC ENVIRONMENTS

Hillel J. Chiel          Randall D. Beer          Leon S. Sterling
Biology Dept.    Dept. of Computer  Engineering and Science and    CS Dept.
& CAISR      Center for Automation and Intelligent Systems Research      & CAISR
CWRU               Case Western Reserve University                CWRU
                      Cleveland, OH 44106

## ABSTRACT

Research in artificial neural networks has generally emphasized homogeneous architectures. In contrast, the nervous systems of natural animals exhibit great heterogeneity in both their elements and patterns of interconnection. This heterogeneity is crucial to the flexible generation of behavior which is essential for survival in a complex, dynamic environment. It may also provide powerful insights into the design of artificial neural networks. In this paper, we describe a heterogeneous neural network for controlling the walking of a simulated insect. This controller is inspired by the neuroethological and neurobiological literature on insect locomotion. It exhibits a variety of statically stable gaits at different speeds simply by varying the tonic activity of a single cell. It can also adapt to perturbations as a natural consequence of its design.

## INTRODUCTION

Even very simple animals exhibit a dazzling variety of complex behaviors which they continuously adapt to the changing circumstances of their environment. Nervous systems evolved in order to generate appropriate behavior in dynamic, uncertain situations and thus insure the survival of the organisms containing them. The function of a nervous system is closely tied to its structure. Indeed, the heterogeneity of nervous systems has been found to be crucial to those few behaviors for which the underlying neural mechanisms have been worked out in any detail [Selverston, 1988]. There is every reason to believe that this conclusion will remain valid as more complex nervous systems are studied:

> The brain as an "organ" is much more diversified than, for example, the kidney or the liver. If the performance of relatively few liver cells is known in detail, there is a good chance of defining the role of the whole organ. In the brain, different cells perform different, specific tasks. . . Only rarely can aggregates of neurons be treated as though they were homogeneous. Above all, the cells in the brain are connected with one another according to a complicated but specific design that is of far greater complexity than the connections between cells in other organs. ([Kuffler, Nicholls, & Martin, 1984], p. 4)

In contrast to research on biological nervous systems, work in artificial neural networks has primarily emphasized uniform networks of simple processing units with a regular interconnection scheme. These homogeneous networks typically depend upon some general learning procedure to train them to perform specific tasks. This approach has certain advantages. Such networks are analytically tractable and one can often prove theorems about their behavior. Furthermore, such networks have interesting computational properties with immediate practical applications. In addition, the necessity of training these networks has resulted in a resurgence of interest in learning, and new training procedures are being developed. When these procedures succeed, they allow the rapid construction of networks which perform difficult tasks.

However, we believe that the role of learning may have been overemphasized in artificial neural networks, and that the architectures and heterogeneity of biological nervous systems have been unduly neglected. We may learn a great deal from more careful study of the design of biological nervous systems and the relationship of this design to behavior. Toward this end, we are exploring the ways in which the architecture of the nervous systems of simpler organisms can be utilized in the design of artificial neural networks. We are particularly interested in developing neural networks capable of continuously synthesizing appropriate behavior in dynamic, underspecified, and uncertain environments of the sort encountered by natural animals.

## THE ARTIFICIAL INSECT PROJECT

In order to address these issues, we have begun to construct a simulated insect which we call *Periplaneta computatrix*. Our ultimate goal is to design a nervous system capable of endowing this insect with all of the behaviors required for long-term survival in a complex and dynamic simulated environment similar to that of natural insects. The skills required to survive in this environment include the basic abilities to move around, to find and consume food when necessary, and to escape from predators. In this paper, we focus on the design of that portion of the insect's nervous system which controls its locomotion.

In designing this insect and the nervous system which controls it, we are inspired by the biological literature. It is important to emphasize, however, that this is not a modeling project. We are not attempting to reproduce the experimental data on a particular animal; rather, we are using insights gleaned from Biology to design neural networks capable of generating similar behaviors. In this manner, we hope to gain a better understanding of the role heterogeneity plays in the generation of behavior by nervous systems, and to abstract design principles for use in artificial neural networks.

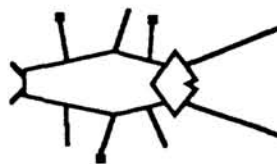

**Figure 1.** *Periplaneta computatrix*

## BODY

The body of our artificial insect is shown in Figure 1. It is loosely based on the American Cockroach, *Periplaneta americana* [Bell & Adiyodi, 1981]. However, it is a reasonable abstraction of the bodies of most insects. It consists of an abdomen, head, six legs with feet, two antennae, and two cerci in the rear. The mouth can open and close and contains tactile and chemical sensors. The antennae also contain tactile and chemical sensors. The cerci contain tactile and wind sensors. The feet may be either up or down. When a foot is down, it appears as a black square. Finally, a leg can apply forces which translate and rotate the body whenever its foot is down.

In addition, though the insect is only two-dimensional, it is capable of "falling down." Whenever its center of mass falls outside of the polygon formed by its supporting feet, the insect becomes statically unstable. If this condition persists for any length of time, then we say that the insect has "fallen down" and the legs are no longer able to move the body.

## NEURAL MODEL

The essential challenge of the Artificial Insect Project is to design neural controllers capable of generating the behaviors necessary to the insect's survival. The neural model that we are currently using to construct our controllers is shown in Figure 2. It represents the firing frequency of a cell as a function of its input potential. We have used saturating linear threshold functions for this relationship (see inset). The RC characteristics of the cell membrane are also represented. These cells are interconnected by weighted synapses which can cause currents to flow through this membrane. Finally, our model includes the possibility of additional intrinsic currents which may be time and voltage dependent. These currents allow us to capture some of the intrinsic properties which make real neurons unique and have proven to be important components of the neural mechanisms underlying many behaviors.

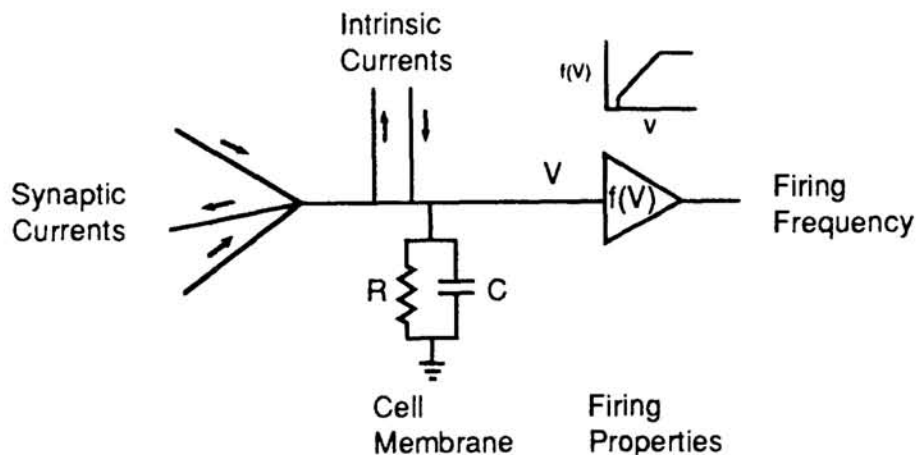

**Figure 2.** Neural Model

For example, a *pacemaker cell* is a neuron which is capable of endogenously producing rhythmic bursting. Pacemakers have been implicated in a number of temporally patterned behaviors and play a crucial role in our locomotion controller. As described by Kandel (1976, pp. 260-268), a pacemaker cell exhibits the following characteristics: (1) when it is sufficiently inhibited, it is silent, (2) when it is sufficiently excited, it bursts continuously, (3) between these extremes, the interburst interval is a continuous function of the membrane potential, (4) a transient excitation which causes the cell to fire between bursts can reset the bursting rhythm, and (5) a transient inhibition which prematurely terminates a burst can also reset the bursting rhythm.

These characteristics can be reproduced with our neural model through the addition of two intrinsic currents. $I_H$ is a depolarizing current which tends to pull the membrane potential above threshold. $I_L$ is a hyperpolarizing current which tends to pull the membrane potential below threshold. These currents change according to the following rules: (1) $I_H$ is triggered whenever the cell goes above threshold or $I_L$ terminates, and it then remains active for a fixed period of time, and (2) $I_L$ is triggered whenever $I_H$ terminates, and it then remains active for a variable period of time whose duration is a function of the membrane potential. In our work to date, the voltage dependence of $I_L$ has been linear.

## LOCOMOTION

An animal's ability to move around its environment is fundamental to many of its other behaviors. In most insects, this requirement is fulfilled by six-legged walking. Thus, this was the first capability we sought to provide to *P. computatrix*. Walking involves the generation of temporally patterned forces and stepping movements such that the insect maintains a steady forward motion at a variety of speeds without falling down. Though we do not address all of these issues here, it is worth pointing out that locomotion is an interesting adaptive behavior in its own right. An insect robustly solves this complex coordination problem in real time in the presence of variations in load and terrain, developmental changes, and damage to the walking apparatus itself [Graham, 1985].

## LEG CONTROLLER

The most basic components of walking are the rhythmic movements of each individual leg. These consist of a *swing phase*, in which the foot is up and the leg is swinging forward, and a *stance phase*, in which the foot is down and the leg is swinging back, propelling the body forward. In our controller, these rhythmic movements are produced by the leg controller circuit shown in Figure 3. There is one command neuron, C, for the entire controller and six copies of the remainder of this circuit, one for each leg.

The rhythmic leg movements are primarily generated centrally by the portion of the leg controller shown in solid lines in Figure 3. Each leg is controlled by three motor neurons. The stance and swing motor neurons determine the force with which the leg is swung backward or forward, respectively, and the foot motor neuron controls whether the foot is up or down. Normally, the foot is down and the stance motor neuron is active, pushing

the leg back and producing a stance phase. Periodically, however, this state is interrupted by a burst from the pacemaker neuron P. This burst inhibits the foot and stance motor neurons and excites the swing motor neuron, lifting the foot and swinging the leg forward. When this burst terminates, another stance phase begins. Rhythmic bursting in P thus produces the basic swing/stance cycle required for walking. The force applied during each stance phase as well as the time between bursts in P depend upon the level of excitation supplied by the command neuron C. This basic design is based on the flexor burst-generator model of cockroach walking [Pearson, 1976].

In order to properly time the transitions between the swing and stance phases, the controller must have some information about where the legs actually are. The simplest way to provide this information is to add sensors which signal when a leg has reached an extreme forward or backward angle, as shown with dashed lines in Figure 3. When the leg is all the way back, the backward angle sensor encourages P to initiate a swing by exciting it. When the leg is all the way forward, the forward angle sensor encourages P to terminate the swing by inhibiting it. These sensors serve to reinforce and fine-tune the centrally generated stepping rhythm. They were inspired by the hair plate receptors in *P. americana*, which seem to play a similar role in its locomotion [Pearson, 1976].

The RC characteristics of our neural model cause delays at the end of each swing before the next stance phase begins. This pause produces a "jerky" walk which we sought to avoid. In order to smooth out this effect, we added a *stance reflex* comprised of the dotted connections shown in Figure 3. This reflex gives the motor neurons a slight "kick" in the right direction to begin a stance whenever the leg is swung all the way forward and is also inspired by the cockroach [Pearson, 1976].

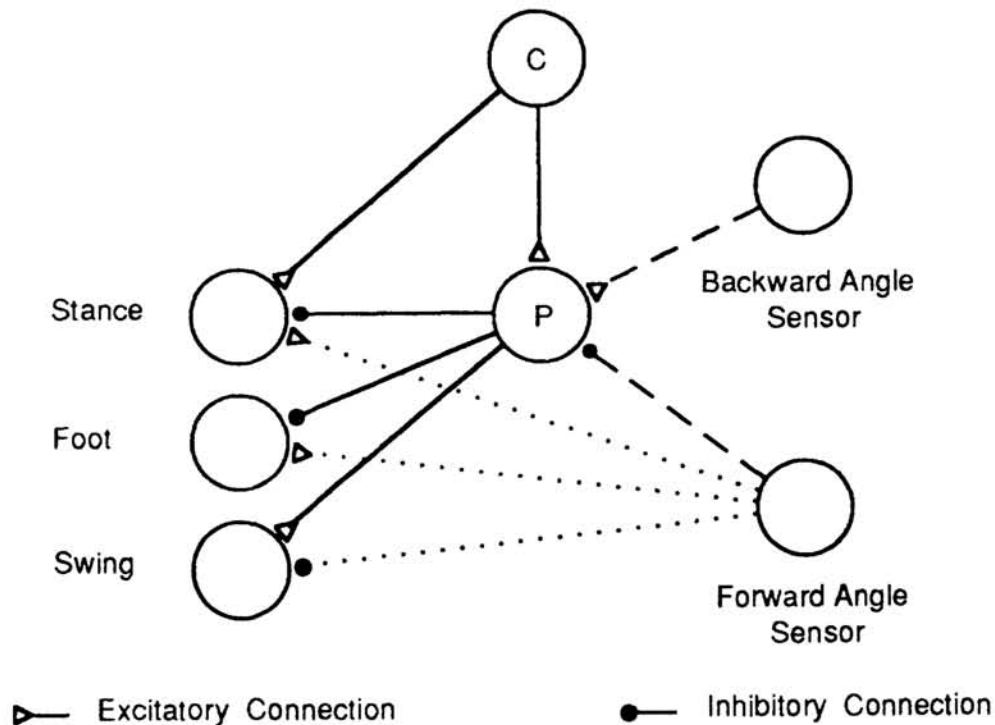

**Figure 3.** Leg Controller Circuit

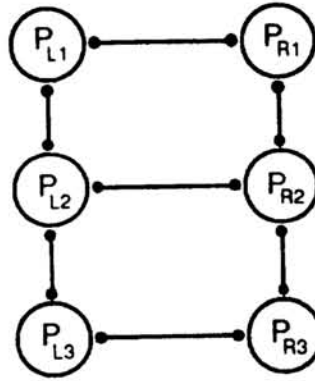

**Figure 4**. Central Coupling between Pacemakers

## LOCOMOTION CONTROLLER

In order for these six individual leg controllers to serve as the basis for a locomotion controller, we must address the issue of stability. Arbitrary patterns of leg movements will not, in general, lead to successful locomotion. Rather, the movements of each leg must be synchronized in such a way as to continuously maintain stability.

A good rule of thumb is that adjacent legs should be discouraged from swinging at the same time. As shown in Figure 4, this constraint was implemented by mutual inhibition between the pacemakers of adjacent legs. So, for example, when leg $L_2$ is swinging, legs $L_1$, $L_3$ and $R_2$ are discouraged from also swinging, but legs $R_1$ and $R_3$ are unaffected (see Figure 5a for leg labelings). This coupling scheme is also derived from Pearson's (1976) work.

The gaits adopted by the controller described above depend in general upon the initial angles of the legs. To further enhance stability, it is desirable to impose some reliable order to the stepping sequence. Many animals exhibit a stepping sequence known as a *metachronal wave*, in which a wave of stepping progresses from back to front. In insects, for example, the back leg swings, then the middle one, then the front one on each side of the body. This sequence is achieved in our controller by slightly increasing the leg angle ranges of the rear legs, lowering their stepping frequency. Under these conditions, the rear leg oscillators entrain the middle and front ones, and produce metachronal waves [Graham, 1977].

## RESULTS

When this controller is embedded in the body of our simulated insect, it reliably produces successful walking. We have found that the insect can be made to walk at different speeds with a variety of gaits simply by varying the firing frequency of the command neuron C. Observed gaits range from the *wave gait*, in which the metachronal waves on each side of the body are very nearly separated, to the *tripod gait*, in which the front and back legs on each side of the body step with the middle leg on the opposite side. These gaits fall out of the interaction between the dynamics of the neural controller and the body in which it is embedded.

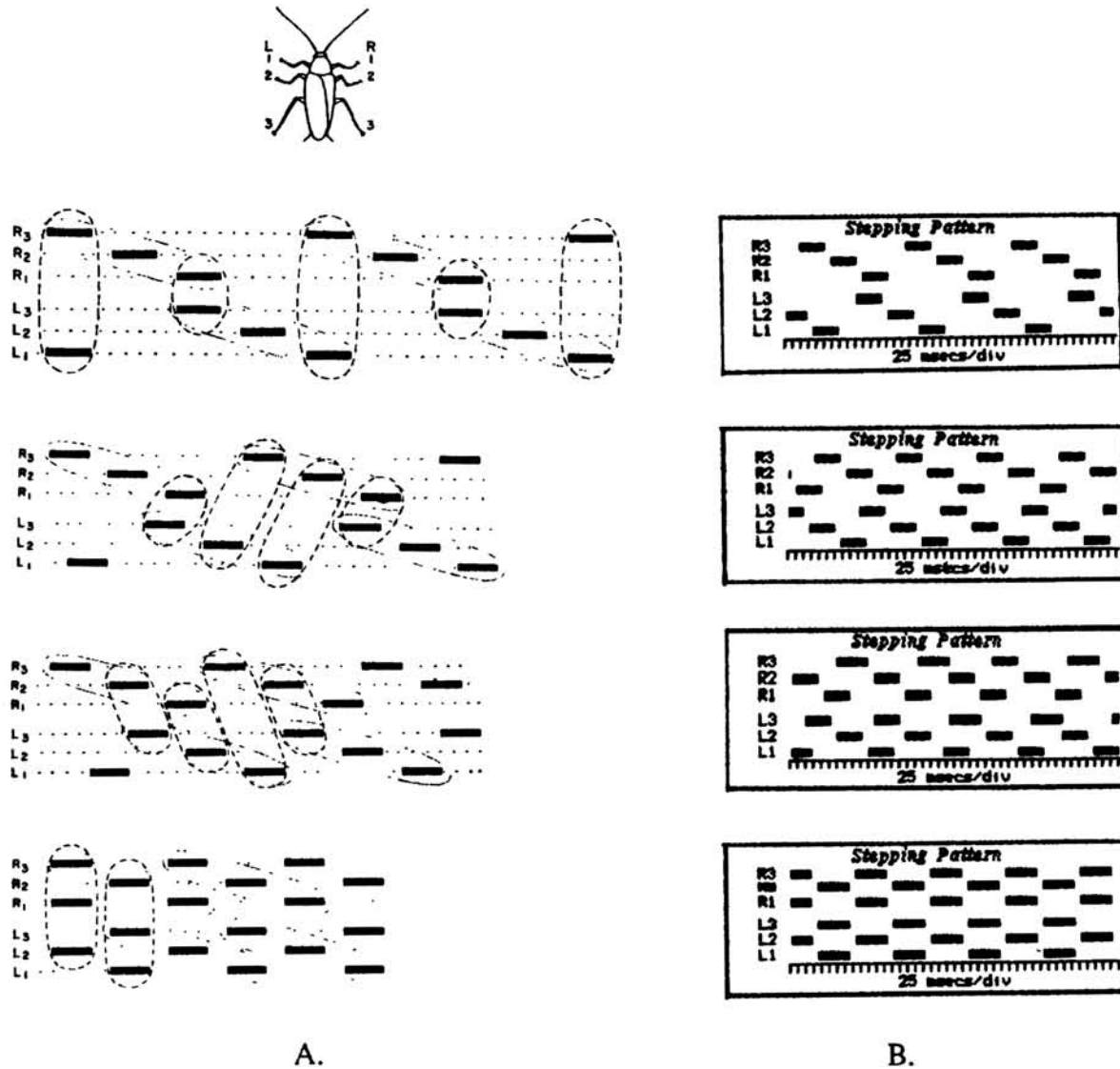

A.                                                    B.

**Figure 5.** (A) Description of Some Gaits Observed in Natural Insects (from [Wilson, 1966]). (B) Selected Gaits Observed in *P. computatrix*.

If the legs are labeled as shown at the top of Figure 5a, then gaits may be conveniently described by their stepping patterns. In this representation, a black bar is displayed during the swing phase of each leg. The space between bars represents the stance phase. Selected gaits observed in *P. computatrix* at different speeds are shown in Figure 5b as the command neuron firing frequency is varied from lowest (top) to highest (bottom). At the lower speeds, the metachronal waves on each side of the body are very apparent. The metachronal waves can still be discerned in faster walks. However, they increasingly overlap as the stance phases shorten, until the tripod gait appears at the highest speeds. This sequence of gaits bears a strong resemblance to some of those that have been described for natural insects, as shown in Figure 5a [Wilson, 1966].

In order to study the robustness of this controller and to gain insight into the detailed mechanisms of its operation, we have begun a series of lesion studies. Such studies ex-

amine the behavioral effects of selective damage to a neural controller. This study is still in progress and we only report a few preliminary results here. In general, we have been repeatedly surprised by the intricacy of the dynamics of this controller. For example, removal of all of the forward angle sensors resulted in a complete breakdown of the metachronal wave at low speeds. However, at higher speeds, the gait was virtually unaffected. Only brief periods of instability caused by the occasional overlap of the slightly longer than normal swing phases were observed in the tripod gait, but the insect did not fall down. Lesioning single forward angle sensors often dynamically produced compensatory phase shifts in the other legs. Lesions of selected central connections produced similarly interesting effects. In general, our studies seem to suggest subtle interactions between the central and peripheral components of the controller which deserve much more exploration.

Finally, we have observed the phenomena of *reflex stepping* in *P. computatrix*. When the central locomotion system is completely shut down by strongly inhibiting the command neuron and the insect is continuously pushed from behind, it is still capable of producing an uncoordinated kind of walking. As the insect is pushed forward, a leg whose foot is down bends back until the backward angle sensor initiates a swing by exciting the pacemaker neuron P. When the leg has swung all the way forward, the stance reflex triggered by the forward angle sensor puts the foot down and the cycle repeats.

Brooks (1989) has described a semi-distributed locomotion controller for an insect-like autonomous robot. We are very much in agreement with his general approach. However, his controller is not as fully distributed as the one described above. It relies on a central leg lift sequencer which must be modified to produce different gaits . Donner (1985) has also implemented a distributed hexapod locomotion controller inspired by an early model of Wilson's (1966). His design used individual leg controllers driven by leg load and position information. These leg controllers were coupled by forward excitation from posterior legs. Thus, his stepping movements were produced by reflex-driven peripheral oscillators rather than the central oscillators used in our model. He did not report the generation of the series of gaits shown in Figure 5a. Donner also demonstrated the ability of his controller to adapt to a missing leg. We have experimented with leg amputations as well, but with mixed success. We feel that more accurate three-dimensional load information than we currently model is necessary for the proper handling of amputations. Neither of these other locomotion controllers utilize neural networks.

## CONCLUSIONS AND FUTURE WORK

We have described a heterogeneous neural network for controlling the walking of a simulated insect. This controller is completely distributed yet capable of reliably producing a range of statically stable gaits at different walking speeds simply by varying the tonic activity of a single command neuron. Lesion studies have demonstrated that the controller is robust, and suggested that subtle interactions and dynamic compensatory mechanisms are responsible for this robustness.

This controller is serving as the basis for a number of other behaviors. We have already implemented wandering, and are currently experimenting with controllers for recoil re-

sponses and edge following. In the near future, we plan to implement feeding behavior and an escape response, resulting in what we feel is the minimum complement of behaviors necessary for survival in an insect-like environment. Finally, we wish to introduce plasticity into these controllers so that they may better adapt to the exigencies of particular environments. We believe that learning is best viewed as a means by which additional flexibility can be added to an existing controller.

The locomotion controller described in this paper was inspired by the literature on insect locomotion. The further development of *P. computatrix* will continue to draw inspiration from the neuroethology and neurobiology of simpler natural organisms. In trying to design autonomous organisms using principles gleaned from Biology, we may both improve our understanding of natural nervous systems and discover design principles of use to the construction of artificial ones. A robot with "only" the behavioral repertoire and adaptability of an insect would be an impressive achievement indeed. In particular, we have argued in this paper for a more careful consideration of the intrinsic architecture and heterogeneity of biological nervous systems in the design of artificial neural networks. The locomotion controller we have described above only hints at how productive such an approach can be.

## References

Bell, W.J. and K.G. Adiyodi eds (1981). *The American Cockroach*. New York: Chapman and Hall.

Brooks, R.A. (1989). A robot that walks: emergent behaviors from a carefully evolved network. *Neural Computation* 1(1).

Donner, M. (1987). *Real-time control of walking* (*Progress in Computer Science, Volume 7*). Cambridge, MA: Birkhäuser Boston, Inc.

Graham, D. (1977). Simulation of a model for the coordination of leg movements in free walking insects. *Biological Cybernetics* 26:187-198.

Graham, D. (1985). Pattern and control of walking in insects. *Advances in Insect Physiology* 18:31-140.

Kandel, E.R. (1976). *Cellular Basis of Behavior: An Introduction to Behavioral Neurobiology*. W.H. Freeman.

Kuffler, S.W., Nicholls, J.G., and Martin, A. R. (1984). *From Neuron to Brain: A Cellular Approach to the Function of the Nervous System*. Sunderland, MA: Sinauer Associates Inc.

Pearson, K. (1976). The control of walking. *Scientific American* 235:72-86.

Selverston, A.I. (1988). A consideration of invertebrate central pattern generators as computational data bases. *Neural Networks* 1:109-117.

Wilson, D.M. (1966). Insect walking. *Annual Review of Entomology* 11:103-122.